# Linear Program Approximations for Factored Continuous-State Markov Decision Processes

**Milos Hauskrecht and Branislav Kveton**
Department of Computer Science and Intelligent Systems Program
University of Pittsburgh
{*milos,bkveton*}*@cs.pitt.edu*

## Abstract

Approximate linear programming (ALP) has emerged recently as one of the most promising methods for solving complex factored MDPs with finite state spaces. In this work we show that ALP solutions are not limited only to MDPs with finite state spaces, but that they can also be applied successfully to factored continuous-state MDPs (CMDPs). We show how one can build an ALP-based approximation for such a model and contrast it to existing solution methods. We argue that this approach offers a robust alternative for solving high dimensional continuous-state space problems. The point is supported by experiments on three CMDP problems with 24-25 continuous state factors.

## 1  Introduction

Markov decision processes (MDPs) offer an elegant mathematical framework for representing and solving decision problems in the presence of uncertainty. While standard solution techniques, such as value and policy iteration, scale-up well in terms of the number of states, the state space of more realistic MDP problems is factorized and thus becomes exponential in the number of state components. Much of the recent work in the AI community has focused on factored structured representations of finite-state MDPs and their efficient solutions. Approximate linear programming (ALP) has emerged recently as one of the most promising methods for solving complex factored MDPs with discrete state components. The approach uses a linear combination of local feature functions to model the value function. The coefficients of the model are fit using linear program methods. A number of refinements of the ALP approach have been developed over past few years. These include the work by Guestrin et al [8], de Farias and Van Roy [6, 5], Schuurmans and Patrascu [15], and others [11]. In this work we show how the same set of linear programming (LP) methods can be extended also to solutions of factored continuous-state MDPs.[1]

The optimal solution of the continuous-state MDP (CMDP) may not (and typically does not) have a finite support. To address this problem, CMDPs and their solutions are usually approximated and solved either through state space discretization or by fitting a surrogate and (often much simpler) parametric value function model. The two methods come with different advantages and limitations. [2] The disadvantage of discretizations is their accu-

racy and the fact that higher accuracy solutions are paid for by the exponential increase in the complexity of discretizations. On the other hand, parametric value-function approximations may become unstable when combined with the dynamic programming methods and least squares error [1]. The ALP solution that is developed in this work eliminates the disadvantages of discretization and function approximation approaches while preserving their good properties. It extends the approach of Trick and Zin [17] to factored multi-dimensional continuous state spaces. Its main benefits are good running time performance, stability of the solution, and good quality policies.

Factored models offer a more natural and compact way of parameterizing complex decision processes. However, not all CMDP models and related factorizations are equally suitable also for the purpose of optimization. In this work we study factored CMDPs with state spaces restricted to $[0, 1]^n$. We show that the solution for such a model can be approximated by an ALP with infinite number of constraints that decompose locally. In addition, we show that by choosing transition models based on beta densities (or their mixtures) and basis functions defined by products of polynomials one obtains an ALP in which both the objective function and constraints are in closed form. In order to alleviate the problem of infinite number of constraints, we develop and study approximation based on constraint sampling [5, 6]. We show that even under a relatively simple random constraint sampling we are able to very quickly calculate solutions of a high quality that are comparable to other existing CMDP solution methods.

The text of the paper is organized as follows. First we review finite-state MDPs and approximate linear programming (ALP) methods developed for their factored refinements. Next we show how to extend the LP approximations to factored continuous-state MDPs and discuss assumptions underlying the model. Finally, we test the new method on a continuous-state version of the computer network problem [8, 15] and compare its performance to alternative CMDP methods.

## 2  Finite-state MDPs

A finite state MDP defines a stochastic control process with components $(X, A, P, R)$, where $X$ is a finite set of states, $A$ is a finite set of actions, $P : X \times A \times X \rightarrow [0, 1]$ defines a probabilistic transition model mapping a state to the next states given an action, and $R : X \times A \rightarrow \mathbf{R}$ defines a reward model for choosing an action in a specific state.

Given an MDP our objective is to find the policy $\pi^* : X \rightarrow A$ maximizing the infinite-horizon discounted reward criterion: $E(\sum_{i=0}^{\infty} \gamma^i r_i)$, where $\gamma \in [0, 1)$ is a discount factor and $r_i$ is a reward obtained in step $i$. The value of the optimal policy satisfies the Bellman fixed point equation [12]:

$$V(\mathbf{x}) = \max_a \left[ R(\mathbf{x}, a) + \gamma \sum_{\mathbf{x}'} P(\mathbf{x}'|\mathbf{x}, a)V(\mathbf{x}') \right], \tag{1}$$

where $V$ is the value of the optimal policy and $\mathbf{x}'$ denotes the next state. For all states $\mathbf{x} \in X$ the equation can be written as $V = HV$, where $H$ is the Bellman operator. Given the value function $V$, the optimal policy $\pi^*(\mathbf{x})$ is defined by the action optimizing Eqn 1.

Methods for solving an MDP include value iteration, policy iteration, and linear programming [12, 2]. In the linear program (LP) formulation we solve the following problem:

$$\text{minimize} \quad \sum_{\mathbf{x}} V(\mathbf{x}) \tag{2}$$

$$\text{subject to:} \quad V(\mathbf{x}) - \gamma \sum_{\mathbf{x}'} P(\mathbf{x}'|\mathbf{x}, a)V(\mathbf{x}') - R(\mathbf{x}, a) \geq 0, \quad \forall \ \mathbf{x}, a$$

where values of $V(\mathbf{x})$ for every state $\mathbf{x}$ are treated as variables.

**Factorizations and LP approximations**

In factored MDPs, the state space $X$ is defined in terms of state variables $\{X_1, X_2, \ldots, X_n\}$. As a result, the state space becomes exponential in the number of variables. Compact parameterizations of MDPs based on dynamic belief networks [7] and decomposable reward functions are routinely used to represent such MDPs more efficiently. However, the presence of a compact model does not imply the existence of efficient optimal solutions. To address this problem Koller and Parr [9] and Guestrin at al [8] propose to use a linear model [13]:

$$f(\mathbf{x}) = \sum_i w_i f_i(\mathbf{x}_i)$$

to approximate the value function $V(\mathbf{x})$. Here $w_i$ are the linear coefficients to be found (fit) and $f_i$s denote feature functions defined over subsets $\mathbf{x}_i$ of state variables.

Given a factored binary-state MDP, the coefficients of the linear model can be found by solving the surrogate of the LP in Equation 2 [8]:

$$\text{minimize}_{\mathbf{w}} \quad \sum_i w_i 2^{n - |\mathbf{x}_i|} \sum_{\mathbf{x}_i} f_i(\mathbf{x}_i) \tag{3}$$

$$\text{subject to:} \quad \sum_i w_i \left[ f_i(\mathbf{x}_i) - \gamma \sum_{\mathbf{x}_i'} P(\mathbf{x}_i' | \mathbf{x}_{i,a}, a) f_i(\mathbf{x}_i') \right] - R(\mathbf{x}, a) \geq 0, \ \forall \mathbf{x}, a$$

where $\mathbf{x}_{i,a}$ are the parents of state variables in $\mathbf{x}_i'$ under action $a$, and $R(\mathbf{x}, a)$ decomposes to $\sum_{r=1}^{m} R_{a,r}(\mathbf{x}_{a,r}, a)$, such that $R_{a,r}(\mathbf{x}_{a,r}, a)$ is a local reward function defined over a subset of state variables. Note that while the objective function can be computed efficiently, the number of constraints one has to satisfy remains exponential in the number of random variables. However, only a subset of these constraints becomes active and affect the solution. Guestrin et al [8] showed how to find active constraints by solving a cost network problem. Unfortunately, the cost network formulation is NP-hard. An alternative approach for finding active constraints was devised by Schuurmans and Patrascu [15]. The approach implements a constraint generation method [17] and appears to give a very good performance on average. The idea is to greedily search for maximally violated constraints which can be done efficiently by solving a linear optimization problem. These constraints are included in the linear program and the process is repeated until no violated constraints are found. De Farias and Van Roy [5] analyzed a Monte Carlo approach with randomly sampled constraints.

## 3 Factored continuous-state MDPs

Many stochastic controlled processes are more naturally defined using continuous state variables. In this work we focus on continuous-state MDPs (CMDPs) where state spaces are restricted to $[0,1]^n$. [3] We assume factored representations where transition probabilities are defined in terms of densities over $[0,1]$ state variable subspaces: $p(\mathbf{x}'|\mathbf{x}, a) = \prod_{j=1}^{n} p(x_j'|\mathbf{x}, a)$ where $\mathbf{x}'$ and $\mathbf{x}$ denote the current and previous states. Rewards are represented compactly over subsets of state variables, similarly to factored finite-state MDPs.

### 3.1 Solving continuous-state MDP

The optimal value function for a continuous-state MDP satisfies the Bellman fixed point equation:

$$V(\mathbf{x}) = \max_a \left[ R(\mathbf{x}, a) + \gamma \int_{\mathbf{x}'} p(\mathbf{x}'|\mathbf{x}, a) V(\mathbf{x}') d\mathbf{x}' \right].$$

The problem with CMDPs is that in most cases the optimal value function does not have a finite support and cannot be computed. The solutions attempt to replace the value function or the optimal policy with a finite approximation.

**Grid-based MDP (GMDP) discretizations.** A typical solution is to discretize the state space to a set of grid points and approximate value functions over such points. Unfortunately, classic grid algorithms scale up exponentially with the number of state variables [4]. Let $G = \{\mathbf{x}^1, \mathbf{x}^2, \ldots, \mathbf{x}^N\}$ be a set of grid points over the state space $[0, 1]^n$. Then the Bellman operator $H$ can be approximated with an operator $H_G$ that is restricted to grid points $G$. One such operator has been studied by Rust [14] and is defined as:

$$V_G(\mathbf{x}^i) = \max_a \left[ R(\mathbf{x}^i, a) + \gamma \sum_{j=1}^N P_G(\mathbf{x}^j | \mathbf{x}^i, a) V_G(\mathbf{x}^j) \right], \qquad (4)$$

where $P_G(\mathbf{x}^j | \mathbf{x}^i, a) = \psi_a(\mathbf{x}^i) p(\mathbf{x}^j | \mathbf{x}^i, a)$ defines a normalized transition probability such that $\psi_a(\mathbf{x}^i)$ is a normalizing constant. Equation 4 applied to grid points $G$ defines a finite state MDP with $|G|$ states. The solution, $V_G = H_G V_G$, approximates the original continuous-state MDP. Convergence properties of the approximation scheme in Equation 4 for random or pseudo-random samples were analyzed by Rust [14].

**Parametric function approximations.** An alternative way to solve a continuous-state MDP is to approximate the optimal value function $V(\mathbf{x})$ with an appropriate parametric function model [3]. The parameters of the model are fitted iteratively by applying one step Bellman backups to a finite set of state points arranged on a fixed grid or obtained through Monte Carlo sampling. Least squares criterion is used to fit the parameters of the model. In addition to parallel updates and optimizations, on-line update schemes based on gradient descent [3, 16] are very popular and can be used to optimize the parameters. The disadvantage of the methods is their instability and possible divergence [1].

### 3.2 LP approximations of CMDPs

Our objective is to develop an alternative to the above solutions that is based on ALP techniques and that takes advantage of model factorizations. It is easy to see that for a general continuous-state model the exact solution cannot be formulated as a linear program as was done in Equation 2 since the number of states is infinite. However, using linear representations of the value functions we need to optimize only over a finite number of weights combining feature functions. So adopting the ALP approach from factored MDPs (Section 2), the CMDP problem can be formulated as:

$$\text{minimize}_{\mathbf{w}} \quad \sum_i w_i \int_{\mathbf{x}_i} f_i(\mathbf{x}_i) d\mathbf{x}_i$$

$$\text{subject to:} \quad \sum_i w_i \left[ f_i(\mathbf{x}_i) - \gamma \int_{\mathbf{x}_i'} \left( \prod_{x_j' \in \mathbf{x}_i'} p(x_j' | \mathbf{x}_{j,a}, a) \right) f_i(\mathbf{x}_i') d\mathbf{x}_i' \right] - R(\mathbf{x}, a) \geq 0, \ \ \forall \ \mathbf{x}, a$$

The above formulation of the ALP builds upon our observation that linear models in combination with factored transitions are well-behaved when integrated over $[0, 1]^n$ state space (or any bounded space) and nicely decompose along state-variable subsets defining feature functions similarly to Equation 3. This simplification is a consequence of the following variable elimination transformation:

$$\int_0^1 \left( \int_z f(z) dz \right) dy = \left[ \left( \int_z f(z) dz \right) y \right]_0^1 = \int_z f(z) dz.$$

Despite the decomposition, the ALP formulation of the factored CMDP comes with two concerns. First, the integrals may be improper and not computable. Second, we need to

satisfy infinite number of constraints (for all values of $\mathbf{x}$ and $a$). In the following we give solutions to both issues.

**Closed form solutions** Integrals in the objective function and constraints depend on the choice of transition models and basis functions. We want all these integrals to be proper Riemannian integrals. We prefer integrals with closed-form expressions. To this point, we have identified conjugate classes of transition models and basis functions leading to closed form expressions.

**Beta transitions.** To parameterize the transition model over $[0, 1]$ we propose to use beta densities or their mixtures. The beta transition is defined as:

$$p(x_j'|\mathbf{x}_{j,a}, a) = Beta(x_j'|g_{j,a}^1(\mathbf{x}_{j,a}), g_{j,a}^2(\mathbf{x}_{j,a})),$$

where $\mathbf{x}_{j,a}$ is the parent set of a variable $x_j$ under action $a$, and $g_{j,a}^1(\mathbf{x}_{j,a}), g_{j,a}^2(\mathbf{x}_{j,a}) > 0$ for $\mathbf{x}_{j,a} \in [0, 1]^{|\mathbf{x}_{j,a}|}$ define the parameters of the beta model.

**Feature functions.** A feature function form that is particularly suitable for the ALP and matches beta transitions is a product of power functions:

$$f_i(\mathbf{x}_i) = \prod_{x_j \in \mathbf{x}_i} x_j^{m_{j,i}}.$$

It is easy to show that for such a case the integrals in the objective function simplify to:

$$\int_{\mathbf{x}_i} f_i(\mathbf{x}_i)d\mathbf{x}_i = \int_{\mathbf{x}_i} \prod_{x_j \in \mathbf{x}_i} x_j^{m_{j,i}}d\mathbf{x}_i = \prod_{x_j \in \mathbf{x}_i} \int_{x_j} x_j^{m_{j,i}}dx_j = \prod_{x_j \in \mathbf{x}_i} \frac{1}{m_{j,i} + 1}.$$

Similarly, using our conjugate transition and basis models the integrals in constraints simplify to:

$$\int_{\mathbf{x}_i'} \left( \prod_{x_j' \in \mathbf{x}_i'} p(x_j'|\mathbf{x}_{j,a}, a) \right) f_i(\mathbf{x}_i')d\mathbf{x}_i' = \prod_{x_j' \in \mathbf{x}_i'} \frac{\Gamma(g_{j,a}^1(\mathbf{x}_{j,a}) + g_{j,a}^2(\mathbf{x}_{j,a}))\Gamma(g_{j,a}^1(\mathbf{x}_{j,a})) + m_{j,i})}{\Gamma(g_{j,a}^1(\mathbf{x}_{j,a}) + g_{j,a}^2(\mathbf{x}_{j,a}) + m_{j,i})\Gamma(g_{j,a}^1(\mathbf{x}_{j,a}))},$$

where $\Gamma(.)$ is the gamma function. For example, assuming features with products of state variables: $f_i(\mathbf{x}_i) = \prod_{x_j \in \mathbf{x}_i} x_j$, the ALP formulation becomes:

$$\text{minimize}_{\mathbf{w}} \quad \sum_i w_i \left(\frac{1}{2}\right)^{|\mathbf{x}_i|} \tag{5}$$

$$\text{subject to:} \quad \sum_i w_i \left[ \prod_{x_j \in \mathbf{x}_i} x_j - \gamma \prod_{x_j' \in \mathbf{x}_i'} \frac{g_{j,a}^1(\mathbf{x}_{j,a})}{g_{j,a}^1(\mathbf{x}_{j,a}) + g_{j,a}^2(\mathbf{x}_{j,a})} \right] - R(\mathbf{x}, a) \geq 0, \quad \forall \mathbf{x}, a$$

**ALP solution.** Although the ALP uses infinitely many constraints, only a finite subset of constraints, active constraints, is necessary to define the optimal solution. Existing ALP methods for factored finite-state MDPs search for this subset more efficiently by taking advantage of local constraint decompositions and various heuristics. However, at the end these methods always rely on the fact the decompositions are defined on a finite state subspace. Unfortunately, constraints in our model decompose over smaller but still continuous subspaces, so the existing solutions for the finite-state MDPs cannot be applied directly.

**Sampling constraints.** To avoid the problem of continuous state spaces we approximate the ALP solution using a finite set of constraints defined by a finite set of state space points and actions in $A$. These state space points can be defined by regular grids on state subspaces or via random sampling of states $\mathbf{x} \in \mathbf{X}$. In this work we focus on and experiment

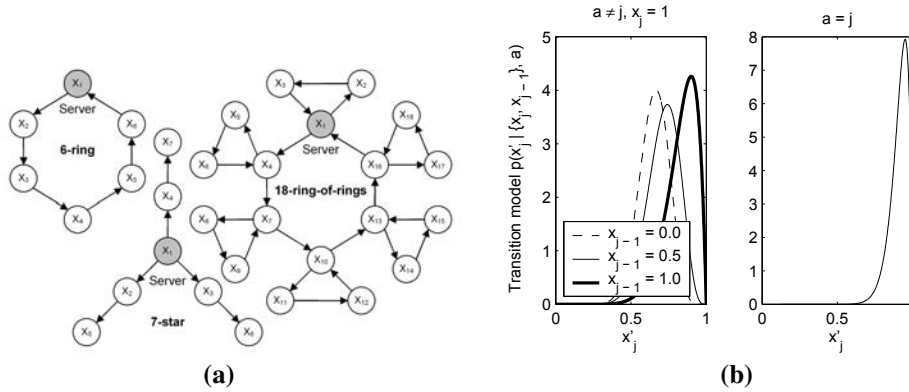

(a)                                        (b)

Figure 1: **a.** Topologies of computer networks used in experiments. **b.** Transition densities for the $j$th computer and different previous-state/action combinations.

with the random sampling approach. For the finite state spaces such a technique has been devised and analyzed by de Farias and Van Roy [5]. We note that the blind sampling approach can be improved via various heuristics.[4] However, despite many possible heuristic improvements, we believe that the crucial benefit comes from the ALP formulation that "fits" the linear model and subsequent constraint and subspace decompositions.

## 4    Experiments

To test the ALP method we use a continuous-state modification of the computer network example proposed by Guestrin et al [8]. Figure 1a illustrates three different network structures used in experiments. Nodes in graphs represent computers. The state of a machine is represented by a number between 0 and 1 reflecting its processing capacity (the ability to process tasks). The network performance can be controlled through activities of a human operator: the operator can attend a machine (one at time) or do nothing. Thus, there is a total of $n + 1$ actions where $n$ is the number of computers in the network. The processing capacity of a machine fluctuates randomly and is determined by: (1) a random event (e.g., a software bug), (2) machines connected to it and (3) the presence of the operator at the machine console. The transition model represents the dynamics of the computer network. The model is factorized and defined in terms of beta densities: $p(x'_j | \mathbf{x}_{j,a}, a) = Beta(x'_j | g^1_{j,a}(\mathbf{x}_{j,a}), g^2_{j,a}(\mathbf{x}_{j,a}))$, where $x'_j$ is the current state of the $j$th computer, and $\mathbf{x}_{j,a}$ describes the previous-step state of the computers affecting $j$. We use: $g^1_{j\neq a,a}(\mathbf{x}_{j,a}) = 2 + 13x_j - 5x_j x_{j-1}$ and $g^2_{j\neq a,a}(\mathbf{x}_{j,a}) = 10 - 2x_j - 6x_j x_{j-1}$ for transitions when the human does not attend the computer, and $g^1_{j=a,a}(\mathbf{x}_{j,a}) = 20$ and $g^2_{j=a,a}(\mathbf{x}_{j,a}) = 2$ when the operator is present at the computer. Figure 1b illustrates transition densities for the $j$th computer given different values of its parents $\{x_j, x_{j-1}\}$ and actions. The goal is to maintain the processing ability of the network at the highest possible level over time. The preferences are expressed in the reward function: $R(\mathbf{x}, a) = 2x_1^2 + \sum_{j=2}^n x_j^2$, where $x_1$ is the server. The discount factor $\gamma$ is $0.95$.

To define the ALP approximation, we used a linear combination of linear (for every node) and quadratic (for every link) feature functions. To demonstrate the practical benefit of the approach we have compared it to the grid-based approximation (Equation 4) and least-square value iteration approach (with the same linear value function model as in the ALP). The constraints in the ALP were sampled randomly. To make the comparison fair the same sets of samples were shared by all three methods. The full comparison study was run on

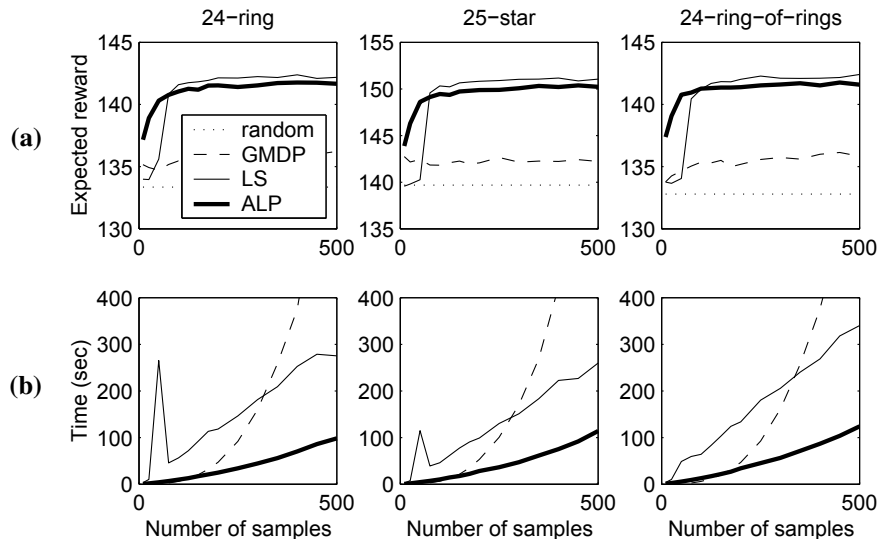

Figure 2: (a) Average values of control policies for ALP, least-squares (LS), and grid (GMDP) approaches for different sample sizes. Random policy is used as a baseline. (b) Average running times.

problems with three network structures from Figure 1a, each with 24 or 25 nodes. Figure 2a illustrates the average quality (value) of a policy obtained by different approximation methods while varying the number of samples. The average is computed over 30 solutions obtained for 30 different sample sets and 100 different (random) start states. The simulation trajectories of length 50 are used. Figure 2b illustrates the scale-up potential of the methods in terms of running times. Results are averaged over 30 solutions.

Overall, the results of experiments clearly demonstrate the benefit of the ALP with "local" feature functions. For the sample size range tested, our ALP method came close to the least-squares (LS) approach in terms of the quality. Both used the same value function model and both managed to fit well the parameters, hence we got comparable quality results. However, the ALP was much better in terms of running time. Oscillations and poor convergence behavior of the iterative LS method is responsible for the difference. The ALP outperformed the grid-based approach (GMDP) in both the policy quality and running times. The gap in the policy quality was more pronounced for smaller sample sizes. This can be explained by the ability of the model to "cover" complete state space as opposed to individual grid points. Better running times for the ALP can be explained by the fact that the number of free variables to be optimized is fixed (they are equal to weights $\mathbf{w}$), while in grid methods free variables correspond to grid samples and their number grows linearly.

## 5  Conclusions

We have extended the application of linear program approximation methods and their benefits to factored MDPs with continuous states. [5] We have proposed a factored transition model based on beta densities and identified feature functions that match well such a model. Our ALP solution offers numerous advantages over standard grid and function approximation approaches: (1) it takes advantage of the structure of the process; (2) it allows one to define non-linear value function models and avoids the instabilities associated with least-squared approximations; (3) it gives a more robust solution for small sample sizes when

compared to grid methods and provides a better way of "smoothing" value function to unseen examples; (4) its running time scales up better than grid methods. These has been demonstrated experimentally on three large problems.

Many interesting issues related to the new method remain to be addressed. First, the random sampling of constraints can be improved using various heuristics. We report results of some heuristic solutions in a separate work [10]. Second, we did not give any complexity bounds for the random constraint sampling approach. However, we expect that the proofs by de Farias and Van Roy [5] can be adapted to cover the CMDP case. Finally, our ALP method assumes a bounded subspace of $I\!R^n$. The important open question is how to extend the ALP method to $I\!R^n$ spaces.

## Footnotes

[1]We assume that action spaces stay finite. Rust [14] calls such models discrete decision processes.

[2]The two methods are described in more depth in Section 3.

[3]We note that in general any bounded subspace of $\mathbf{R}^n$ can be transformed to $[0,1]^n$.

[4]Various constraint sampling heuristics are analyzed and reported in a separate work [10].

[5]We note that our CMDP solution paves the road to ALP solutions for factored hybrid state MDPs.

# References

[1] D.P. Bertsekas. A counter-example to temporal differences learning. *Neural Computation*, 7:270–279, 1994.

[2] D.P. Bertsekas. *Dynamic programming and optimal control*. Athena Scientific, 1995.

[3] D.P. Bertsekas and J.N. Tsitsiklis. *Neuro-dynamic Programming*. Athena Sc., 1996.

[4] C.S. Chow and J.N. Tsitsiklis. An optimal one-way multigrid algorithm for discrete-time stochastic control. *IEEE Transactions on Automatic Control*, 36:898–914, 1991.

[5] D. P. de Farias and B. Van Roy. On constraint sampling for the linear programming approach to approximate dynamic programming. *Mathematics of Operations Research*, submitted, 2001.

[6] D.P. de Farias and B. Van Roy. The Linear Programming Approach to Approximate Dynamic Programming. In *Operations Research*, 51:6, 2003.

[7] T. Dean and K. Kanazawa. A model for reasoning about persistence and causation. *Computational Intelligence*, 5:142–150, 1989.

[8] C. Guestrin, D. Koller, and R. Parr. Max-norm projections for factored MDPs. In *Proceedings of the Seventeenth International Joint Conference on Artificial Intelligence*, pages 673–682, 2001.

[9] D. Koller and R. Parr. Computing factored value functions for policies in structured MDPs. In *Proceedings of the 16th International Joint Conference on Artificial Intelligence*, pages 1332–1339, 1999.

[10] B. Kveton and M. Hauskrecht. Heuristics refinements of approximate linear programming for factored continuous-state Markov decision processes. In *14Th International Conference on Automated Planning and Scheduling*, to appear, 2004.

[11] P. Poupart, C. Boutilier, R. Patrascu, and D. Schuurmans. Piecewise linear value function approximation for factored MDPs. In *Proceedings of the Eighteenth National Conference on AI*, pages 292–299, 2002.

[12] M.L. Puterman. *Markov decision processes: discrete stochastic dynamic programming*. John Wiley, New York, 1994.

[13] B. Van Roy. *Learning and value function approximation in complex decision problems*. PhD thesis, Massachussetts Institute of Technology, 1998.

[14] J. Rust. Using randomization to break the course of dimensionality. *Econometrica*, 65:487–516, 1997.

[15] D. Schuurmans and R.Patrascu. Direct value-approximation for factored MDPs. In *Advances in Neural Information Processing Systems 14*, MIT Press, 2002.

[16] R. S. Sutton and A. G. Barto. *Reinforcement Learning: An introduction*. 1998.

[17] M. Trick and E.S Zin. A linear programming approach to solving stochastic dynamic programs, TR, 1993.
